# Exact learning curves for Gaussian process regression on large random graphs

**Matthew J. Urry**
Department of Mathematics
King's College London
London, WC2R 2LS, U.K.
matthew.urry@kcl.ac.uk

**Peter Sollich**
Department of Mathematics
King's College London
London, WC2R 2LS, U.K.
peter.sollich@kcl.ac.uk

## Abstract

We study learning curves for Gaussian process regression which characterise performance in terms of the Bayes error averaged over datasets of a given size. Whilst learning curves are in general very difficult to calculate we show that for discrete input domains, where similarity between input points is characterised in terms of a graph, accurate predictions can be obtained. These should in fact become exact for large graphs drawn from a broad range of random graph ensembles with arbitrary degree distributions where each input (node) is connected only to a finite number of others. Our approach is based on translating the appropriate belief propagation equations to the graph ensemble. We demonstrate the accuracy of the predictions for Poisson (Erdos-Renyi) and regular random graphs, and discuss when and why previous approximations of the learning curve fail.

## 1 Introduction

Learning curves are a convenient way of characterising the performance that can be achieved with machine learning algorithms: they give the generalisation error $\epsilon$ as a function of the number of training examples $n$, averaged over all datasets of size $n$ under appropriate assumptions about the data-generating process. Such a characterization is particularly useful in the case of non-parametric approaches such as Gaussian processes (GPs) [1], where in contrast to the parametric case [2] there is no generic classification of possible learning curves.

Here we study GP regression, where a real-valued output function $f(x)$ is to be learned. Qualitatively, GP learning curves are relatively well understood for the scenario where the inputs $x$ come from a continuous space, typically $\mathbb{R}^n$ [3, 4, 5, 6, 7, 8, 9, 10, 11]. However, except in the limit of large $n$, or for very specific situations like one-dimensional inputs [3], the learning curves cannot be calculated exactly. Here we show that this *is* possible for discrete input spaces where similarity between input points can be represented as a graph whose edges connect similar points, inspired by work at last year's NIPS that developed simple approximations for this scenario [12].

There are many potential application domains where learning of such functions of discrete inputs $x$ could be relevant, for example if $x$ is a research paper whose impact $f(x)$ we would like to predict; the similarity graph could then be constructed on the basis of shared authorship. Or we could be trying to learn functions on generic symbol strings $x$, for example ones characterizing protein amino acid sequences, and the similarity graph would have edges between homologous molecules.

Our aim is to find out how well GP regression can perform in such discrete domains; alternative inference approaches including online algorithms [13, 14, 15, 16] would also be interesting to study but are outside the scope of the present paper. We focus on large sparse random graphs, where each node is connected only to a finite number of other nodes even though the overall number of nodes in the graph is large.

In section 2 we give a brief overview of GP regression and summarize the approximation for the learning curves used in previous work [4, 8, 12]. Section 3 then explains our method: following a similar approach in [17] for random matrix spectra, we write down the belief propagation equations for a given graph in the form normally used in the cavity method [18] of statistical mechanics, and then translate them to graphs drawn from a random graph ensemble. Because for sparse random graphs typical loop lengths grow with the graph size, the belief propagation equations and hence our learning curve predictions should become exact for large graphs.

Section 4 compares the predictions with simulation results for Poisson (Erdos-Renyi) graphs, where each edge is independently present with some small probability, and random regular graphs, where each node has the same degree (number of neighbours). The new predictions are indeed very accurate, and substantially more so than previous approximations. In section 4.1 we discuss in more detail the relationship between our work and these approximations to rationalize where the strongest deviations occur. Finally, section 5 summarises our results and discusses open questions and directions for future work.

## 2   GP regression and approximate learning curves

Gaussian processes have become a well known machine learning technique used in a wide range of areas, see e.g. [19, 20, 21]. One reason for their success is the intuitive way that a priori information about the function to be learned is transparently encoded by the covariance and mean functions of the GP.

A GP is a Gaussian prior over functions $f$ with a fixed covariance function (kernel) $C$ and mean function (assumed to be $0$)[1]. In the simplest case the likelihood is also Gaussian, i.e. we assume that the outputs $y_\mu$ in a set of examples $D = \{(i_1, y_1, \ldots, (i_N, y_N)\}$ are obtained by corrupting the clean function values $f_{i_\mu}$ with i.i.d. Gaussian noise of variance $\sigma^2$. Then the posterior distribution over functions is, from Bayes' theorem $P(f|D) \propto P(f)P(D|f)$:

$$P(f|D) \propto \exp(-\frac{1}{2}\boldsymbol{f}^T\boldsymbol{C}^{-1}\boldsymbol{f} - \frac{1}{2\sigma^2}\sum_{\mu=1}^{N}(y_\mu - f_{i_\mu})^2) \tag{1}$$

We consider GPs in discrete spaces, where each input is a node of a graph and can therefore be given a discrete label $i$ as anticipated above; $f_i$ is the associated function value. If the graph has $V$ nodes, the covariance function is then just a $V \times V$ matrix.

A number of possible forms for covariance functions on graphs have been proposed. We will focus on the relatively flexible random walk covariance function [22],

$$\boldsymbol{C} = \frac{1}{\kappa}((1 - a^{-1})\boldsymbol{I} + a^{-1}\boldsymbol{D}^{-1/2}\boldsymbol{A}\boldsymbol{D}^{-1/2})^p \qquad a \geq 2, \quad p \geq 0 \tag{2}$$

Here $\boldsymbol{A}$ is the adjacency matrix of the graph, with $A_{ij} = 1$ if nodes $i$ and $j$ are connected by an edge, and 0 otherwise; $\boldsymbol{D} = \text{diag}\{d_1, \ldots, d_V\}$ is a diagonal matrix containing the degrees of the nodes in the graph ($d_i = \sum_j A_{ij}$). One can easily see the relationship to a random walk: the unnormalised covariance function is a (symmetrised) $p$-step 'lazy' random walk, with probability $a^{-1}$ of moving to a neighbouring node at each step. The prior thus assumes that function values up to a distance $p$ along the graph are correlated with each other, to an extent determined by the hyperparameter $a^{-1}$. The constant $\kappa$ will be chosen throughout to normalise $\boldsymbol{C}$ so that $\frac{1}{V}\sum_i C_{ii} = 1$, which corresponds to setting the average prior variance of the function values to unity.

Our main concern in this paper are GP learning curves in discrete input spaces. The learning curve describes how the average generalisation error (mean square error) $\epsilon$ decreases with the number of examples $N$. Qualitatively, it gives the rate at which one would expect a GP to learn a function in the *average case*. The generalisation error on an ensemble of graphs is given by

$$\epsilon = \langle \frac{1}{V}\sum_i(\bar{f}_i - f_i)^2 \rangle_{f|D,D,\text{graphs}} \tag{3}$$

where $\boldsymbol{f}$ is the uncorrupted (clean) teacher or target function, and $\bar{\boldsymbol{f}}$ is the posterior mean function of the GP which gives the function values we predict on the basis of the data $D$. It is worth noting that the generalisation error for a graph ensemble contains an additional average over this ensemble. As is standard in the study of learning curves we have assumed a matched scenario where the posterior $P(f|D)$ for our predictions is also the posterior over the underlying target functions. The generalisation error is then the Bayes error, and is given by the average posterior variance.

Sollich [4] and later Opper [7] with a more general replica approach showed that for continuous input spaces a reasonable approximation to the learning curve could be expressed as the solution of the following self-consistent equation:

$$\epsilon = g\left(\frac{N}{\epsilon + \sigma^2}\right), \qquad g(h) = \sum_{\alpha=1}^{V}(\lambda_\alpha^{-1} + h)^{-1} \tag{4}$$

Here the $\lambda_\alpha$ are appropriately defined eigenvalues of the covariance function. The motivation for our study is work presented at NIPS2009 [12], which demonstrated that this approximation can also be used in discrete domains, but is not always accurate. Studying random walk and diffusion kernels [22] on random regular graphs, the authors showed that although the eigenvalue-based approximation is reasonable for both the large and the small $N$ limits, it fails to accurately predict the learning curve in the important transition region between these two extremes, drastically so for low noise variances $\sigma^2$.

In the next section we will show that this shortcoming can be overcome by the cavity method (belief propagation) which explicitly takes advantage of the sparse structure of the underlying graph. This will give an accurate approximation for the learning curves in a broad range of ensembles of sparse random graphs.

## 3   Accurate predictions with the cavity method

The cavity method was developed in statistical physics [18] but is closely related to belief propagation; for a good overview of these and other mean field methods, see e.g. [23]. We begin with equation (3). Because we only need the posterior variance in the matched case considered here, we can shift $\boldsymbol{f}$ so that $\bar{\boldsymbol{f}} = \boldsymbol{0}$; $f_i$ is then the deviation of the function value at node $i$ from the posterior mean. In this notation, the Bayes error is

$$\epsilon = \langle \frac{1}{V}\sum_i \int d\boldsymbol{f} f_i^2 P(f|D)\rangle_{D,\text{graphs}} \tag{5}$$

where $P(f|D)$ now contains in the exponent only the terms from (1) that are quadratic in $\boldsymbol{f}$.

To set up the cavity method, we begin by defining a *generating* or *partition function* $Z$, for a fixed graph, as

$$Z = \int d\boldsymbol{f} \exp(-\frac{1}{2}\boldsymbol{f}^T\boldsymbol{C}^{-1}\boldsymbol{f} - \frac{1}{2\sigma^2}\sum_\mu f_{i_\mu}^2 - \frac{\lambda}{2}\sum_i f_i^2) \tag{6}$$

An auxiliary parameter $\lambda$ has been added here to allow us to represent the Bayes error as $\epsilon = -\lim_{\lambda \to 0}(2/V)\frac{\partial}{\partial \lambda}\langle \log Z\rangle_{D,\text{graphs}}$. The dependence on the dataset $D$ appears in $Z$ only through the sum over $\mu$. It will be more useful to write this as a sum over all nodes: if $n_i$ counts the number of examples seen at node $i$, then $\sum_\mu f_{i_\mu}^2 = \sum_i n_i f_i^2$. Even with this replacement, the partition function in equation (6) is not yet suitable for an application of the cavity method since the inverse covariance function cannot be written explicitly and generates interaction terms $f_i f_j$ between nodes that can be far away from each other along the graph. To eliminate the inverse of the covariance function we therefore perform a Fourier transform on the first term in the exponent, $\exp(-\frac{1}{2}\boldsymbol{f}^T\boldsymbol{C}^{-1}\boldsymbol{f}) \propto \int d\boldsymbol{h} \exp(-\frac{1}{2}\boldsymbol{h}^T\boldsymbol{C}\boldsymbol{h} + i\sum_i h_i f_i)$. The integral over $\boldsymbol{f}$ then factorizes over the $f_i$, and one finds

$$Z \propto \int d\boldsymbol{h} \exp(-\frac{1}{2}\boldsymbol{h}^T\boldsymbol{C}\boldsymbol{h} - \frac{1}{2}\boldsymbol{h}^T\text{diag}\{(\frac{n_i}{\sigma^2} + \lambda)^{-1}\}\boldsymbol{h}) \tag{7}$$

Substituting the explicit form of the covariance function (2) into equation (7) we have

$$Z \propto \int d\boldsymbol{h} \exp(-\frac{1}{2}\boldsymbol{h}^T \sum_{q=0}^p c_q (\boldsymbol{D}^{-1/2}\boldsymbol{A}\boldsymbol{D}^{-1/2})^q \boldsymbol{h} - \frac{1}{2}\boldsymbol{h}^T\text{diag}\{(\frac{n_i}{\sigma^2} + \lambda)^{-1}\}\boldsymbol{h}) \tag{8}$$

where we have written the power in equation (2) as a binomial sum and defined $c_q = p!/[q!(p-q)!]a^{-q}(1-a^{-1})^{p-q}/\kappa$.

For $p > 1$, equation (8) still has interactions with more than the immediate neighbours. To solve this we introduce additional variables $\boldsymbol{h}^q$, defined recursively via $\boldsymbol{h}^q = (\boldsymbol{D}^{-1/2}\boldsymbol{A}\boldsymbol{D}^{-1/2})\boldsymbol{h}^{q-1}$ for $q \geq 1$ and $\boldsymbol{h}^0 = \boldsymbol{h}$. These definitions are enforced via Dirac delta-functions, each $i$ and $q \geq 1$ giving a factor $\delta(h_i^q - d_i^{-1/2}\sum_j A_{ij}d_j^{-1/2}h_j^{q-1}) \propto \int d\hat{h}_i^q \exp[i\hat{h}_i^q(h_i^q - d_i^{-1/2}\sum_j A_{ij}d_j^{-1/2}h_j^{q-1})]$. Substituting this into equation (8) gives the key advantage that now the adjacency matrix appears only linearly in the exponent, so that we have interactions only across edges of the graph. Rescaling the $h_i^q$ to $d_i^{1/2}h_i^q$ and similarly for the $\hat{h}_i^q$, and explicitly separating off the local terms from the interactions finally yields

$$Z \propto \int \prod_{q=0}^{p} d\boldsymbol{h}^q \prod_{q=1}^{p} d\hat{\boldsymbol{h}}^q \prod_i \exp(-\frac{1}{2}\sum_{q=0}^{p} c_q d_i h_i^0 h_i^q - \frac{1}{2}\frac{d_i(h_i^0)^2}{n_i/\sigma^2 + \lambda} + i\sum_{q=1}^{p} d_i \hat{h}_i^q h_i^q)$$
$$\times \prod_{(ij)} \exp(-i\sum_{q=1}^{p}(\hat{h}_i^q h_j^{q-1} + \hat{h}_j^q h_i^{q-1})) \tag{9}$$

We now have the partition function of a (complex-valued) Gaussian graphical model. By differentiating $\log Z$ with respect to $\lambda$, keeping track of $\lambda$-dependent prefactors not written above, one finds that the Bayes error is,

$$\epsilon = \lim_{\lambda \to 0} \frac{1}{V}\sum_i \frac{1}{n_i/\sigma^2 + \lambda}\left(1 - \frac{d_i\langle (h_i^0)^2\rangle}{n_i/\sigma^2 + \lambda}\right) \tag{10}$$

and so we need the marginal distributions of the $h_i^0$. This is where the cavity method enters: for a large random graph the structure is locally treelike, so that if node $i$ were eliminated the corresponding subgraphs (locally trees) rooted at the neighbours $j \in \mathcal{N}(i)$ of $i$ would become independent [17]. The resulting cavity marginals $P_j^{(i)}(\boldsymbol{h}_j, \hat{\boldsymbol{h}}_j | D)$ can then be calculated iteratively within these subgraphs, giving the cavity update equations

$$P_j^{(i)}(\boldsymbol{h}_j, \hat{\boldsymbol{h}}_j | D) \propto \exp(-\frac{1}{2}\sum_{q=0}^{p} c_q d_j h_j^0 h_j^q - \frac{1}{2}\frac{d_j(h_j^0)^2}{n_j/\sigma^2 + \lambda} + i\sum_{q=1}^{p} d_j \hat{h}_j^q h_j^q)$$
$$\int \prod_{k \in \mathcal{N}(j)\backslash i} d\boldsymbol{h}_k d\hat{\boldsymbol{h}}_k \exp(-i\sum_{q=1}^{p}(\hat{h}_j^q h_k^{q-1} + \hat{h}_k^q h_j^{q-1}))P_k^{(j)}(\boldsymbol{h}_k, \hat{\boldsymbol{h}}_k | D) \tag{11}$$

One sees that these equations are solved self-consistently by complex-valued Gaussian distributions with mean zero and covariance matrices $\boldsymbol{V}_j^{(i)}$. By performing the Gaussian integrals in the cavity update equations (11) explicitly, these equations then take the rather simple form

$$\boldsymbol{V}_j^{(i)} = (\boldsymbol{O}_j - \sum_{k \in \mathcal{N}(j)\backslash i} \boldsymbol{X}\boldsymbol{V}_k^{(j)}\boldsymbol{X})^{-1} \tag{12}$$

where we have defined the $(2p+1) \times (2p+1)$ matrices

$$\boldsymbol{O}_i = d_i \begin{pmatrix} c_0 + \frac{1}{n_i/\sigma^2+\lambda} & \frac{1}{2}c_1 & \cdots & \frac{1}{2}c_p & 0 & \cdots & 0 \\ \frac{1}{2}c_1 & & & & -i & & \\ \vdots & & & & & \ddots & \\ \frac{1}{2}c_p & & & & & & -i \\ \hline 0 & -i & & & & & \\ \vdots & & \ddots & & & \boldsymbol{0}_{p,p} & \\ 0 & & & -i & & & \end{pmatrix}, \quad \boldsymbol{X} = \begin{pmatrix} & & & & i & & \\ & \boldsymbol{0}_{p+1,p+1} & & & & \ddots & \\ & & & & & & i \\ \hline & & & & 0 & \cdots & 0 \\ i & & 0 & & & & \\ & \ddots & & \vdots & & \boldsymbol{0}_{p,p} & \\ & & i & 0 & & & \end{pmatrix}$$

Finally we need to translate these equations to an ensemble of large sparse graphs. Each ensemble is characterised by the distribution $p(d)$ of the degrees $d_i$, with every graph that has the desired degree distribution being assigned the same probability. Instead of individual cavity covariance

matrices $\boldsymbol{V}_j^{(i)}$, we need to consider their probability distribution $W(\boldsymbol{V})$ across all edges of the graph. Picking at random an edge $(i, j)$ of a graph, the probability that node $j$ will have degree $d_j$ is then $p(d_j)d_j/\bar{d}$, because such a node has $d_j$ "chances" of being picked. (The normalisation factor is the average degree $\bar{d}$.) Using again the locally treelike structure, the incoming (to node $j$) cavity covariances $\boldsymbol{V}_k^{(j)}$ will be i.i.d. samples from $W(\boldsymbol{V})$. Thus a fixed point of the cavity update equations corresponds to a fixed point of an update equation for $W(\boldsymbol{V})$:

$$W(\boldsymbol{V}) = \left\langle \sum_d \frac{p(d)d}{\bar{d}} \int \prod_{k=1}^{d-1} d\boldsymbol{V}_k\, W(\boldsymbol{V}_k)\, \delta(\boldsymbol{V} - (\boldsymbol{O} - \sum_{k=1}^{d-1} \boldsymbol{X}\boldsymbol{V}_k\boldsymbol{X})^{-1}) \right\rangle_n \tag{13}$$

Because the node label is now arbitrary, we have abbreviated $\boldsymbol{V}_j^{(i)}$ to $\boldsymbol{V}$, $d_j$ to $d$, $\boldsymbol{O}_j$ to $\boldsymbol{O}$ and $\boldsymbol{V}_k^{(j)}$ to $\boldsymbol{V}_k$. The average is over the distribution over the number of examples $n \equiv n_j$ at node $j$ in the dataset $D$. Assuming for simplicity that examples are drawn with uniform input probability across all nodes, this distribution is simply $n \sim \text{Poisson}(\nu)$ in the limit of large $N$ and $V$ at fixed $\nu = N/V$.

In general equation (13) – which can also be formally derived using the replica approach [24] – cannot be solved analytically, but we can solve it numerically using a standard population dynamics method [25]. Once we have $W(\boldsymbol{V})$, the Bayes error can be found from the graph ensemble version of equation (10), which is obtained by inserting the explicit expression for $\langle (h_i^0)^2 \rangle$ in terms of the cavity marginals of the neighbouring nodes, and replacing the average over nodes with an average over $p(d)$:

$$\epsilon = \lim_{\lambda \to 0} \left\langle \sum_d \frac{p(d)}{n/\sigma^2 + \lambda} \left(1 - \frac{d}{n/\sigma^2 + \lambda} \int \prod_{k=1}^d d\boldsymbol{V}_k\, W(\boldsymbol{V}_k)\, (\boldsymbol{O} - \sum_{k=1}^d \boldsymbol{X}\boldsymbol{V}_k\boldsymbol{X})_{00}^{-1} \right) \right\rangle_n \tag{14}$$

The number of examples at the node is again to be averaged over $n \sim \text{Poisson}(\nu)$. The subscript "00" indicates the top left element of the matrix, which determines the variance of $h^0$.

To be able to use equation (14), it needs to be rewritten in a form that remains explicitly non-singular when $n = 0$ and $\lambda \to 0$. We split off the $n$-dependence of the matrix inverse by writing $\boldsymbol{O} - \sum_{k=1}^d \boldsymbol{X}\boldsymbol{V}_k\boldsymbol{X} = \boldsymbol{M} + [d/(n/\sigma^2 + \lambda)]\boldsymbol{e}_0\boldsymbol{e}_0^T$, where $\boldsymbol{e}_0^T = (1, 0, \dots, 0)$. The matrix inverse appearing above can then be expressed using the Woodbury formula as

$$\boldsymbol{M}^{-1} - \frac{\boldsymbol{M}^{-1}\boldsymbol{e}_0\boldsymbol{e}_0^T\boldsymbol{M}^{-1}}{(n/\sigma^2 + \lambda)/d + \boldsymbol{e}_0^T\boldsymbol{M}^{-1}\boldsymbol{e}_0} \tag{15}$$

To extract the (0,0)-element (top left) as required we multiply by $\boldsymbol{e}_0^T \cdots \boldsymbol{e}_0$. After some simplification the $\lambda \to 0$ limit can then be taken, with the result

$$\epsilon = \left\langle \sum_d p(d) \int \prod_{k=1}^d d\boldsymbol{V}_k\, W(\boldsymbol{V}_k)\, \frac{1}{n/\sigma^2 + d(\boldsymbol{M}^{-1})_{00}} \right\rangle_n \tag{16}$$

This has a simple interpretation: the cavity marginals of the neighbours provide an effective Gaussian prior for each node, whose inverse variance is $d(\boldsymbol{M}^{-1})_{00}$.

The self-consistency equation (13) for $W(\boldsymbol{V})$ and the expression (16) for the resulting Bayes error are our main results. They allow us to predict learning curves as a function of the number of examples per node, $\nu$, for *arbitrary degree distributions* $p(d)$ of our random graph ensemble providing the graphs are sparse, and for arbitrary noise level $\sigma^2$ and covariance function hyperparameters $p$ and $a$.

We note briefly that in graphs with isolated nodes ($d = 0$), one has to be slightly careful as already in the definition of the covariance function (2) one should replace $\boldsymbol{D} \to \boldsymbol{D} + \delta\boldsymbol{I}$ to avoid division by zero, taking $\delta \to 0$ at the end. For $d = 0$ one then finds in the expression (16) that $(\boldsymbol{M}^{-1})_{00} = \frac{1}{c_0\delta}$ so that $(\delta + d)(\boldsymbol{M}^{-1})_{00} = \delta(\boldsymbol{M}^{-1})_{00} = 1/c_0$. This is to be expected since isolated nodes each have a separate Gaussian prior with variance $c_0$.

# 4 Results

We will begin by comparing the performance of our new cavity prediction (equation (16)) against the eigenvalue approximation (equation (4)) from [4, 7], for random regular graphs with degree 3 (so that $p(d) = \delta_{d,3}$). In this way we can exploit the work of [12], where the quality of the approximation (4) for this case was studied in some detail.

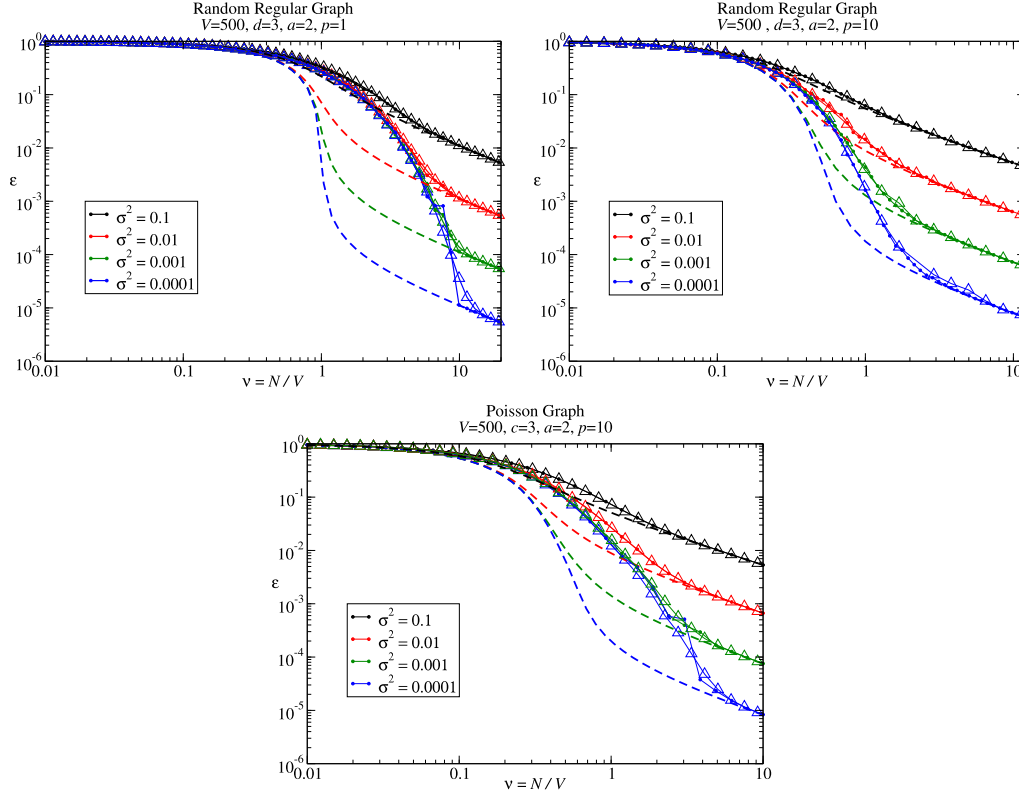

Figure 1: (Left) A comparison of the cavity prediction (solid line with triangles) against the eigenvalue approximation (dashed line) for the learning curves for random regular graphs of degree 3, and against simulation results for graphs with $V = 500$ nodes (solid line with circles). Random walk kernel with $p = 1$, $a = 2$; noise level as shown. (Right) As before with $p = 10$, $a = 2$. (Bottom) Similarly for Poisson (Erdos-Renyi) graphs with $c = 3$.

As can be seen in figure 1 (left) & (right) the cavity approach is accurate along the entire learning curve, to the point where the prediction is visually almost indistinguishable from the numerical simulation results. Importantly, the cavity approach predicts even the midsection of the learning curve for intermediate values of $\nu$, where the eigenvalue prediction clearly fails. The deviations between cavity theory and the eigenvalue predictions are largest in this central part because at this point fluctuations in the number examples seen at each node have the greatest effect. Indeed, for much smaller $\nu$, the dataset does not contain any examples from many of the nodes, i.e. $n = 0$ is dominant and fluctuations towards larger $n$ have low probability. For large $\nu$, the dataset typically contains many examples for each node and Poisson fluctuations around the average value $n = \nu$ are small. The fluctuation effects for intermediate $\nu$ are suppressed when the noise level $\sigma^2$ is large, because then the generalisation error in the range of intermediate $\nu$ is still fairly close to its initial value ($\nu = 0$). But for the smaller noise levels fluctuations in the number of examples for each node can have a large effect, and correspondingly the eigenvalue prediction becomes very poor for intermediate $\nu$. We discuss this further in section 4.1.

Comparing figure 1 (left) and (right), it can also be seen that unlike the eigenvalue-based approximation, the cavity prediction for the learning curve does not deteriorate as $p$ is varied towards lower values. Similar conclusions apply with regard to changes of $a$ (results not shown).

Next we consider Poisson (Erdos-Renyi) graphs, where each edge is present independently with probability $c/V$ [26]. This leads to a Poisson distribution of degrees, $p(d) = e^{-c}c^d/d!$. Figure 1 (bottom) shows the performance of our cavity prediction for this graph ensemble with $c = 3$ for a GP with $p = 10$, $a = 2$, in comparison to simulation results for $V = 500$. The cavity prediction clearly outperforms the eigenvalue-based approximation and again remains accurate even in the central part of the learning curve. Taken together, the results for random regular and Poisson graphs clearly confirm our expectation that the cavity prediction for the learning curve that we have derived should be exact for large graphs. It is worth noting that our new cavity prediction will work for arbitrary degree distributions and is limited only by the assumption of graph sparsity.

## 4.1 Why the eigenvalue approximation fails

The derivation of the eigenvalue approximation (4) by Opper in [8] gives some insight into when and how this approximation breaks down. Opper takes equation (6) and uses the replica trick to write $\langle \log Z \rangle_D = \lim_{n \to 0} \frac{1}{n} \log \langle Z^n \rangle_D$. The average of $Z^n$ is calculated for integer $n$ and then appropriately continued to $n \to 0$. The required $n^{\text{th}}$ power of equation (6) is in our case

$$\langle Z^n \rangle_D = \int \prod_{a=1}^{n} d\boldsymbol{f^a} \langle \exp(-\frac{1}{2} \sum_a \boldsymbol{f}^{aT} \boldsymbol{C}^{-1} \boldsymbol{f}^a - \frac{1}{2\sigma^2} \sum_{i,a} n_i (f_i^a)^2 - \frac{\lambda}{2} \sum_{i,a} (f_i^a)^2) \rangle_D \quad (17)$$

The dataset average, over $n_i \sim \text{Poisson}(\nu)$, then gives

$$\langle Z^n \rangle_D = \int \prod_{a=1}^{n} d\boldsymbol{f^a} \exp(-\frac{1}{2} \sum_a \boldsymbol{f}^{aT} \boldsymbol{C}^{-1} \boldsymbol{f}^a + \nu \sum_i (e^{-\sum_a (f_i^a)^2/2\sigma^2} - 1) - \frac{\lambda}{2} \sum_{i,a} (f_i^a)^2) \quad (18)$$

If one now wants to proceed without explicitly exploiting the sparse graph structure, one has to approximate the exponential term in the exponent. Opper does this using a variational approximation for the distribution of the $\boldsymbol{f^a}$, of Gaussian form, and this eventually leads to the approximation (4) for the learning curve. This approach is evidently justified for large $\sigma^2$, where a Taylor expansion of the exponential term in (18) can be truncated after the quadratic term. For small noise levels, on the other hand, the Gaussian variational approach clearly does not capture all the details of the fluctuations in the numbers of examples $n_i$. By comparison, in this paper, using the cavity method we are able to retain the average over $D$ explicitly, without the need to approximate the distribution of the $n_i$. The result of this is that the section of the learning curve where fluctuations in numbers of examples play a large role is captured accurately, while the Gaussian variational (eigenvalue) approach can give wildly inaccurate results there.

## 5 Conclusions and further work

In this paper we have studied the learning curves of GP regression on large random graphs. In a significant advance on the work of [12], we showed that the approximations for learning curves proposed by Sollich [4] and Opper [7] for continuous input spaces can be greatly improved upon in the graph case, by using the cavity method. We argued that the resulting predictions should in fact become exact in the limit of large random graphs.

Section 3 derived the learning curve approximation using the cavity method for *arbitrary degree distributions*. We defined a *generating function Z* (equation (6)) from which the generalisation error $\epsilon$ can be obtained by differentiation. We then rewrote this using Fourier transforms (equation (7)) and introduced additional variables (equation (9)) to get $Z$ into the required form for a cavity approach: the partition function of a complex-valued Gaussian graphical model. By standard arguments we then derived the cavity update equations for a fixed graph (equation (12)). Finally we generalised from these to graph ensembles (equation (13)), taking the limit of large graph size. The resulting prediction for the generalisation error (equation (16)) has an intuitively appealing interpretation, where each node in the graph learns subject to an effective (and data-dependent) Gaussian prior provided by its neighbours.

In section 4 we compared our new prediction to the eigenvalue approximation results in [12]. We showed that our new method is far more accurate in the challenging midsection of the learning curves than the eigenvalue version, both for random regular and Poisson graph ensembles (figure 1).

Subsection 4.1 discusses why the older approximation, derived from a replica perspective in [7], is inaccurate compared to the cavity method. To retain tractable averages in continuous input spaces, it has to approximate fluctuations in the dataset of the number of examples for each node, thus resulting in the inaccurate predictions seen in figure 1. On graphs one is able to perform this average explicitly when calculating cavity updates and the resulting Bayes error, giving a far more accurate prediction of the learning curves.

Although the learning curves predicted using the cavity method cover a broad range of graph ensembles because they apply for arbitrary $p(d)$, there do remain some interesting types of graph ensembles (for instance graphs with community structure) that cannot be generated by imposing only the degree distribution. Indeed, an important assumption in the current work is that small loops are rare whilst in community graphs, where nodes exhibit preferential attachment, there can be many small loops. We are in the process of analysing GP learning on such graphs using the approach of Rogers *et al.* [27], where community graphs are modelled as having a sparse superstructure joining clusters of densely connected nodes.

Following previous studies [12], we have in this paper set the scale of the covariance function by normalising the average prior covariance over all nodes. For the Poisson graph case our learning curve simulations then show, however, that there can be large variations in the local prior variances $C_{ii}$, while from the Bayesian modelling point of view it would seem more plausible to use covariance functions where all $C_{ii} = 1$. This could be achieved by pre- and post-multiplying the random walk covariance matrix by an appropriate diagonal matrix. We hope to study this modified covariance function in future, and to extend the cavity prediction for the learning curves to this case.

It would also be interesting to expand our approach to model mismatch, where we assume the data-generating process is a GP with hyperparameters that differ from those of the GP being used for inference. This was studied for continuous input spaces in [10]; equally interesting would be a study of mismatch with a fixed target function as analysed by Opper *et al.* [8]. It should further be useful to study the case of mismatched *graphs*, rather than hyperparameters. This is relevant because frequently in real world learning one will have only partial knowledge of the graph structure, for instance in metabolic networks when not all of the pathways have been discovered, or social networks where friendships are continuously being made and broken.

Another interesting avenue for further research would be to look at multiple output (multi-task) GPs on graphs, to see if the work of Chai [28] can be extended to this scenario. One would hope that, as seen with the learning curves for single output GPs in this paper, input domains defined by graphs might allow simplifications in the analysis and provide more accurate bounds or even exact predictions.

Finally, it would be worth extending the study of graph mismatch to the case of evolving graphs and functions. Here spatio-temporal GP regression could be employed to predict functions changing over time, perhaps including a model based approach as in [29] to account for the evolving graph structure.

## Footnotes

[1]We focus on the zero prior mean case throughout. All results translate fairly straightforwardly to the non-zero mean case, but this complicates the algebra without leading to substantially new insights.

## References

[1] Carl E. Rasmussen and Christopher K. I. Williams. *Gaussian Processes for Machine Learning (Adaptive Computation and Machine Learning)*. MIT Press, December 2005.

[2] Shun-ichi Amari, Naotake Fujita, and Shigeru Shinomoto. Four types of learning curves. *Neural Computation*, 4(4):605–618, 1992.

[3] M. Opper. Regression with Gaussian processes: Average case performance. *Theoretical Aspects of Neural Computation: A Multidisciplinary Perspective. Springer-Verlag*, pages 17–23, 1997.

[4] P. Sollich. Learning curves for Gaussian processes. In *Advances in Neural Information Processing Systems 11*, pages 344–350. MIT Press, 1999.

[5] F. Vivarelli and M. Opper. General bounds on Bayes errors for regression with Gaussian processes. In *Advances in Neural Information Processing Systems 11*, pages 302–308. MIT Press, 1999.

[6] C. K. I. Williams and F. Vivarelli. Upper and lower bounds on the learning curve for Gaussian processes. *Machine Learning*, 40(1):77–102, 2000.

[7] M. Opper and D. Malzahn. Learning curves for gaussian processes regression: A framework for good approximations. In *Advances in Neural Information Processing Systems 14*, pages 273–279. MIT Press, 2001.

[8] M. Opper and D. Malzahn. A variational approach to learning curves. In *Advances in Neural Information Processing Systems 14*, pages 463–469. MIT Press, 2002.

[9] P. Sollich and A. Halees. Learning curves for Gaussian process regression: Approximations and bounds. *Neural Computation*, 14(6):1393–1428, 2002.

[10] P. Sollich. Gaussian process regression with mismatched models. In *Advances in Neural Information Processing Systems 14*, pages 519–526. MIT Press, 2002.

[11] P. Sollich. Can Gaussian process regression be made robust against model mismatch? In N Lawrence J Winkler and M Niranjan, editors, *Deterministic and Statistical Methods in Machine Learning*, pages 211–228, Berlin, 2005. Springer.

[12] P. Sollich, M. J. Urry, and C. Coti. Kernels and learning curves for Gaussian process regression on random graphs. In *Advances in Neural Information Processing Systems 22*, pages 1723–1731. Curran Associates, Inc., 2009.

[13] M. Herbster, M. Pontil, and L. Wainer. Online learning over graphs. In *ICML '05: Proceedings of the 22nd international conference on Machine learning*, pages 305–312, New York, NY, USA, 2005. ACM.

[14] M. Herbster and M. Pontil. Prediction on a graph with a perceptron. In *Advances in Neural Information Processing Systems 19*, pages 577–584. MIT Press, 2007.

[15] M. Herbster. Exploiting cluster-structure to predict the labeling of a graph. In *Proceedings of the 19th international conference on Algorithmic Learning Theory*, pages 54–69. Springer, 2008.

[16] M. Belkin, I. Matveeva, and P. Niyogi. Regularization and semi-supervised learning on large graphs. *Learning theory*, 3120:624–638, 2004.

[17] Tim Rogers, Koujin Takeda, Issac Pérez Castillo, and Reimer Kühn. Cavity approach to the spectral density of sparse symmetric random matricies. *Physical Review E*, 78(3):31116–31121, 2008.

[18] M. Mezard, G. Parisi, and M. A. Virasoro. Random free energies in spin glasses. *Le journal de physique - lettres*, 46(6):217–222, 1985.

[19] M. T. Farrell and A. Correa. Gaussian process regression models for predicting stock trends. *Relation*, 10:3414, 2007.

[20] B. Ferris, D. Haehnel, and D. Fox. Gaussian processes for signal strength-based location estimation. In *Proceedings of Robotics: Science and Systems*, Philadelphia, USA, August 2006.

[21] Sunho Park and Seungjin Choi. Gaussian process regression for voice activity detection and speech enhancement. In *International Joint Conference on Neural Networks*, pages 2879–2882, Hong Kong, China, 2008. Institute of Electrical and Electronics Engineers (IEEE).

[22] A. J. Smola and R. Kondor. Kernels and regularization on graphs. In M. Warmuth and B. Scholkopf, editors, *Learning theory and Kernel machines: 16th Annual Conference on Learning Theory and 7th Kernel Workshop (COLT)*, pages 144–158, Heidelberg, 2003. Springer.

[23] M. Opper and D. Saad. *Advanced mean field methods: Theory and practice*. MIT Press, 2001.

[24] Reimer Kühn. Finitely coordinated models for low-temperature phases of amorphous systems. *Journal of Physics A*, 40(31):9227, 2007.

[25] M. Mézard and G. Parisi. The Bethe lattice spin glass revisited. *The European Physical Journal B*, 20(2):217–233, 2001.

[26] P. Erdös and A. Rényi. On random graphs, I. *Publicationes Mathematicae (Debrecen)*, 6:290–297, 1959.

[27] Tim Rogers, Conrad Pérez Vicente, Koujin Takeda, and Isaac Pérez Castillo. Spectral density of random graphs with topological constraints. *Journal of Physics A*, 43(19):195002, 2010.

[28] Kian Ming Chai. Generalization errors and learning curves for regression with multi-task Gaussian processes. In *Advances in Neural Information Processing Systems 22*, pages 279–287. Curran Associates, Inc., 2009.

[29] M. Alvarez, D. Luengo, and N. D. Lawrence. Latent force models. In D. van Dyk and M. Welling, editors, *Proceedings of the Twelfth International Workshop on Artificial Intelligence and Statistics*, pages 9–16, Clearwater Beach, FL, USA, 2009. MIT Press.

